# Learning Multiple Tasks with a Sparse Matrix-Normal Penalty

**Yi Zhang**
Machine Learning Department
Carnegie Mellon University
`yizhang1@cs.cmu.edu`

**Jeff Schneider**
The Robotics Institute
Carnegie Mellon University
`schneide@cs.cmu.edu`

## Abstract

In this paper, we propose a matrix-variate normal penalty with sparse inverse covariances to couple multiple tasks. Learning multiple (parametric) models can be viewed as estimating a matrix of parameters, where rows and columns of the matrix correspond to tasks and features, respectively. Following the matrix-variate normal density, we design a penalty that decomposes the full covariance of matrix elements into the Kronecker product of row covariance and column covariance, which characterizes both task relatedness and feature representation. Several recently proposed methods are variants of the special cases of this formulation. To address the overfitting issue and select meaningful task and feature structures, we include sparse covariance selection into our matrix-normal regularization via $\ell 1$ penalties on task and feature inverse covariances. We empirically study the proposed method and compare with related models in two real-world problems: detecting landmines in multiple fields and recognizing faces between different subjects. Experimental results show that the proposed framework provides an effective and flexible way to model various different structures of multiple tasks.

## 1 Introduction

Learning multiple tasks has been studied for more than a decade [6, 24, 11]. Research in the following two directions has drawn considerable interest: learning a common feature representation shared by tasks [1, 12, 30, 2, 3, 9, 23], and directly inferring the relatedness of tasks [4, 26, 21, 29]. Both have a natural interpretation if we view learning multiple tasks as estimating a matrix of model parameters, where the rows and columns correspond to tasks and features. From this perspective, learning the feature structure corresponds to discovering the structure of the columns in the parameter matrix, and modeling the task relatedness aims to find and utilize the relations among rows.

Regularization methods have shown promising results in finding either feature or task structure [1, 2, 12, 21]. In this paper we propose a new regularization approach and show how several previous approaches are variants of special cases of it. The key contribution is a matrix-normal penalty with sparse inverse covariances, which provides a framework for characterizing and coupling the model parameters of related tasks. Following the matrix normal density, we design a penalty that decomposes the full covariance of matrix elements into the Kronecker product of row and column covariances, which correspond to task and feature structures in multi-task learning. To address overfitting and select task and feature structures, we incorporate sparse covariance selection techniques into our matrix-normal regularization framework via $\ell 1$ penalties on task and feature inverse covariances. We compare the proposed method to related models on two real-world data sets: detecting landmines in multiple fields and recognizing faces between different subjects.

## 2 Related Work

Multi-task learning has been an active research area for more than a decade [6, 24, 11]. For joint learning of multiple tasks, connections need to be established to couple related tasks. One direction is to find a common feature structure shared by tasks. Along this direction, researchers proposed to infer task structure via principal components [1, 12], independent components [30] and covariance [2, 3] in the parameter space, to select a common subset of features [9, 23], as well as to use shared hidden nodes in neural networks [6, 11]. Specifically, learning a shared feature covariance for model parameters [2] is a special case of our proposed framework. On the other hand, assuming models of all tasks are equally similar is risky. Researchers recently began exploring methods to infer the relatedness of tasks. These efforts include using mixtures of Gaussians [4] or Dirichlet processes [26] to model task groups, encouraging clustering of tasks via a convex regularization penalty [21], identifying "outlier" tasks by robust t-processes [29], and inferring task similarity from task-specific features [8, 27, 28]. The present paper uses the matrix normal density and $\ell$1-regularized sparse covariance selection to specify a structured penalty, which provides a systematic way to characterize and select both task and feature structures in multiple parametric models.

Matrix normal distributions have been studied in probability and statistics for several decades [13, 16, 18] and applied to predictive modeling in the Bayesian literature. For example, the standard matrix normal can serve as a prior for Bayesian variable selection in multivariate regression [9], where MCMC is used for sampling from the resulting posterior. Recently, matrix normal distributions have also been used in nonparametric Bayesian approaches, especially in learning Gaussian Processes (GPs) for multi-output prediction [7] and collaborative filtering [27, 28]. In this case, the covariance function of the GP prior is decomposed as the Kronecker product of a covariance over functions and a covariance over examples. We note that the proposed matrix-normal penalty with sparse inverse covariances in this paper can also be viewed as a new matrix-variate prior, upon which Bayesian inference can be performed. We will pursue this direction in our future work.

## 3 Matrix-Variate Normal Distributions

### 3.1 Definition

The matrix-variate normal distribution is one of the most widely studied matrix-variate distributions [18, 13, 16]. Consider an $m \times p$ matrix $\mathbf{W}$. Since we can vectorize $\mathbf{W}$ to be a $mp \times 1$ vector, the normal distribution on a matrix $\mathbf{W}$ can be considered as a multivariate normal distribution on a vector of $mp$ dimensions. However, such an ordinary multivariate distribution ignores the special structure of $\mathbf{W}$ as an $m \times p$ matrix, and as a result, the covariance characterizing the elements of $\mathbf{W}$ is of size $mp \times mp$. This size is usually prohibitive for modeling and estimation. To utilize the structure of $\mathbf{W}$, matrix normal distributions assume that the $mp \times mp$ covariance can be decomposed as the Kronecker product $\boldsymbol{\Sigma} \otimes \boldsymbol{\Omega}$, and elements of $\mathbf{W}$ follow:

$$Vec(\mathbf{W}) \sim N(Vec(\mathbf{M}), \boldsymbol{\Sigma} \otimes \boldsymbol{\Omega}) \tag{1}$$

where $\boldsymbol{\Omega}$ is an $m \times m$ positive definite matrix indicating the covariance between rows of $\mathbf{W}$, $\boldsymbol{\Sigma}$ is a $p \times p$ positive definite matrix indicating the covariance between columns of $\mathbf{W}$, $\boldsymbol{\Sigma} \otimes \boldsymbol{\Omega}$ is the Kronecker product of $\boldsymbol{\Sigma}$ and $\boldsymbol{\Omega}$, $\mathbf{M}$ is a $m \times p$ matrix containing the expectation of each element of $\mathbf{W}$, and $Vec$ is the vectorization operation which maps a $m \times p$ matrix into a $mp \times 1$ vector. Due to the decomposition of covariance as the Kronecker product, the matrix-variate normal distribution of an $m \times p$ matrix $\mathbf{W}$, parameterized by the mean $\mathbf{M}$, row covariance $\boldsymbol{\Omega}$ and column covariance $\boldsymbol{\Sigma}$, has a compact log-density [18]:

$$\log P(\mathbf{W}) = -\frac{mp}{2}\log(2\pi) - \frac{p}{2}\log(|\boldsymbol{\Omega}|) - \frac{m}{2}\log(|\boldsymbol{\Sigma}|) - \frac{1}{2}tr\{\boldsymbol{\Omega}^{-1}(\mathbf{W}-\mathbf{M})\boldsymbol{\Sigma}^{-1}(\mathbf{W}-\mathbf{M})^T\} \tag{2}$$

where $| \ |$ is the determinant of a square matrix, and $tr\{\}$ is the trace of a square matrix.

### 3.2 Maximum likelihood estimation (MLE)

Consider a set of $n$ samples $\{\mathbf{W}_i\}_{i=1}^n$ where each $\mathbf{W}_i$ is a $m \times p$ matrix generated by a matrix-variate normal distribution as eq. (2). The maximum likelihood estimation (MLE) of mean $\mathbf{M}$ is [16]:

$$\hat{\mathbf{M}} = \frac{1}{n}\sum_{i=1}^n \mathbf{W}_i \tag{3}$$

The MLE estimators of $\boldsymbol{\Omega}$ and $\boldsymbol{\Sigma}$ are solutions to the following system:

$$\begin{cases} \hat{\boldsymbol{\Omega}} &=& \frac{1}{np}\sum_{i=1}^{n}(\mathbf{W}_i - \hat{\mathbf{M}})\hat{\boldsymbol{\Sigma}}^{-1}(\mathbf{W}_i - \hat{\mathbf{M}})^T \\ \hat{\boldsymbol{\Sigma}} &=& \frac{1}{nm}\sum_{i=1}^{n}(\mathbf{W}_i - \hat{\mathbf{M}})^T\hat{\boldsymbol{\Omega}}^{-1}(\mathbf{W}_i - \hat{\mathbf{M}}) \end{cases} \quad (4)$$

It is efficient to iteratively solve (4) until convergence, known as the "flip-flop" algorithm [16].

Also, $\hat{\boldsymbol{\Omega}}$ and $\hat{\boldsymbol{\Sigma}}$ are not identifiable and solutions for maximizing the log density in eq. (2) are not unique. If $(\boldsymbol{\Omega}^*, \boldsymbol{\Sigma}^*)$ is an MLE estimate for the row and column covariances, for any $\alpha > 0$, $(\alpha\boldsymbol{\Omega}^*, \frac{1}{\alpha}\boldsymbol{\Sigma}^*)$ will lead to the same log density and thus is also an MLE estimate. This can be seen from the definition in eq. (1), where only the Kronecker product $\boldsymbol{\Sigma} \otimes \boldsymbol{\Omega}$ is identifiable.

# 4 Learning Multiple Tasks with a Sparse Matrix-Normal Penalty

Regularization is a principled way to control model complexity [20]. Classical regularization penalties (for single-task learning) can be interpreted as assuming a multivariate prior distribution on the parameter vector and performing maximum-a-posterior estimation, e.g., $\ell 2$ penalty and $\ell 1$ penalty correspond to multivariate Gaussian and Laplacian priors, respectively. For multi-task learning, it is natural to use matrix-variate priors to design regularization penalties.

In this section, we propose a matrix-normal penalty with sparse inverse covariances for learning multiple related tasks. In Section 4.1 we start with learning multiple tasks with a matrix-normal penalty. In Section 4.2 we study how to incorporate sparse covariance selection into our framework by further imposing $\ell 1$ penalties on task and feature inverse covariances. In Section 4.3 we outline the algorithm, and in Section 4.4 we discuss other useful constraints in our framework.

## 4.1 Learning with a Matrix Normal Penalty

Consider a multi-task learning problem with $m$ tasks in a $p$-dimensional feature space. The training sets are $\{\mathbf{D}_t\}_{t=1}^{m}$, where each set $\mathbf{D}_t$ contains $n_t$ examples $\{(\mathbf{x}_i^{(t)}, y_i^{(t)})\}_{i=1}^{n_t}$. We want to learn $m$ models for the $m$ tasks but appropriately share knowledge among tasks. Model parameters are represented by an $m \times p$ matrix $\mathbf{W}$, where parameters for a task correspond to a row.

The last term in the matrix-variate normal density (2) provides a structure to couple the parameters of multiple tasks as a matrix $\mathbf{W}$: 1) we set $\mathbf{M} = \mathbf{0}$, indicating a preference for simple models; 2) the $m \times m$ row covariance $\boldsymbol{\Omega}$ describes the similarity among tasks; 3) the $p \times p$ column covariance matrix $\boldsymbol{\Sigma}$ represents a shared feature structure. This yields the following total loss $\mathcal{L}$ to optimize:

$$\mathcal{L} = \sum_{t=1}^{m}\sum_{i=1}^{n_t} L(y_i^{(t)}, \mathbf{x}_i^{(t)}, \mathbf{W}(t,:)) + \lambda\, tr\{\boldsymbol{\Omega}^{-1}\mathbf{W}\boldsymbol{\Sigma}^{-1}\mathbf{W}^T\} \quad (5)$$

where $\lambda$ controls the strength of the regularization, $(y_i^{(t)}, \mathbf{x}_i^{(t)})$ is the $i$th example in the training set of the $t$th task, $\mathbf{W}(t,:)$ is the parameter vector of the $t$th task, and $L()$ is a convex empirical loss function depending on the specific model we use, e.g., squared loss for linear regression, log-likelihood loss for logistic regression, hinge loss for SVMs, and so forth. When $\boldsymbol{\Omega}$ and $\boldsymbol{\Sigma}$ are known and positive definite, eq. (5) is convex w.r.t. $\mathbf{W}$ and thus $\mathbf{W}$ can be optimized efficiently [22].

Now we discuss a few special cases of (5) and how is previous work related to them. When we fix $\boldsymbol{\Omega} = \mathbf{I}_m$ and $\boldsymbol{\Sigma} = \mathbf{I}_p$, the penalty term can be decomposed into standard $\ell 2$-norm penalties on the $m$ rows of $\mathbf{W}$. In this case, the $m$ tasks in (5) can be learned almost independently using single-task $\ell 2$ regularization (but tasks are still tied by sharing the parameter $\lambda$).

When we fix $\boldsymbol{\Omega} = \mathbf{I}_m$, tasks are linked only by a shared feature covariance $\boldsymbol{\Sigma}$. This corresponds to a multi-task feature learning framework [2, 3] which optimizes eq. (5) w.r.t. $\mathbf{W}$ and $\boldsymbol{\Sigma}$, with an additional constraint $tr\{\boldsymbol{\Sigma}\} \le 1$ on the trace of $\boldsymbol{\Sigma}$ to avoid setting $\boldsymbol{\Sigma}$ to infinity.

When we fix $\boldsymbol{\Sigma} = \mathbf{I}_p$, tasks are coupled only by a task similarity matrix $\boldsymbol{\Omega}$. This is used in a recent clustered multi-task learning formulation [21], which optimizes eq. (5) w.r.t. $\mathbf{W}$ and $\boldsymbol{\Omega}$, with additional constraints on the singular values of $\boldsymbol{\Omega}$ that are motivated and derived from task clustering. A more recent multi-label classification model [19] essentially optimizes $\mathbf{W}$ in eq. (5) with a label correlation $\boldsymbol{\Omega}$ given as prior knowledge and empirical loss $L$ as the max-margin hinge loss.

We usually do not know task and feature structures in advance. Therefore, we would like to infer $\boldsymbol{\Omega}$ and $\boldsymbol{\Sigma}$ in eq. (5). Note that if we jointly optimize $\mathbf{W}$, $\boldsymbol{\Omega}$ and $\boldsymbol{\Sigma}$ in eq. (5), we will always set $\boldsymbol{\Omega}$ and $\boldsymbol{\Sigma}$ to be infinity matrices. We can impose constraints on $\boldsymbol{\Omega}$ and $\boldsymbol{\Sigma}$ to avoid this, but a more natural way is to further expand eq. (5) to include all relevant terms w.r.t. $\boldsymbol{\Omega}$ and $\boldsymbol{\Sigma}$ from the matrix normal log-density (2). As a result, the total loss $\mathcal{L}$ is:

$$\mathcal{L} = \sum_{t=1}^{m} \sum_{i=1}^{n_t} L(y_i^{(t)}, \mathbf{x}_i^{(t)}, \mathbf{W}(t,:)) + \lambda \left[ p \log |\boldsymbol{\Omega}| + m \log |\boldsymbol{\Sigma}| + tr\{\boldsymbol{\Omega}^{-1} \mathbf{W} \boldsymbol{\Sigma}^{-1} \mathbf{W}^T\} \right] \quad (6)$$

Based on this formula, we can infer task structure $\boldsymbol{\Omega}$ and feature structure $\boldsymbol{\Sigma}$ given the model parameters $\mathbf{W}$, as the following problem:

$$\min_{\boldsymbol{\Omega}, \boldsymbol{\Sigma}} \; p \log |\boldsymbol{\Omega}| + m \log |\boldsymbol{\Sigma}| + tr\{\boldsymbol{\Omega}^{-1} \mathbf{W} \boldsymbol{\Sigma}^{-1} \mathbf{W}^T\} \quad (7)$$

This problem is equivalent to maximizing the log-likelihood of a matrix normal distribution as in eq. (2), given $\mathbf{W}$ as observations and expectation $\mathbf{M}$ fixed at $\mathbf{0}$. Following Section 3.2, the MLE of $\boldsymbol{\Omega}$ and $\boldsymbol{\Sigma}$ can be obtained by the "flip-flop" algorithm:

$$\left\{ \begin{array}{rcl} \hat{\boldsymbol{\Omega}} &=& \frac{1}{p} \mathbf{W} \hat{\boldsymbol{\Sigma}}^{-1} \mathbf{W}^T + \epsilon \mathbf{I}_m \\ \hat{\boldsymbol{\Sigma}} &=& \frac{1}{m} \mathbf{W}^T \hat{\boldsymbol{\Omega}}^{-1} \mathbf{W} + \epsilon \mathbf{I}_p \end{array} \right. \quad (8)$$

where $\epsilon$ is a small positive constant to improve numerical stability. As discussed in Section 3.2, only $\boldsymbol{\Sigma} \otimes \boldsymbol{\Omega}$ is uniquely defined, and $\hat{\boldsymbol{\Omega}}$ and $\hat{\boldsymbol{\Sigma}}$ are only identifiable up to an multiplicative constant. This will not affect the optimization of $\mathbf{W}$ using eq. (5), since only $\boldsymbol{\Sigma} \otimes \boldsymbol{\Omega}$ matters for this purpose.

## 4.2 Sparse Covariance Selection in the Matrix-Normal Penalty

Consider the sparsity of $\boldsymbol{\Omega}^{-1}$ and $\boldsymbol{\Sigma}^{-1}$. When $\boldsymbol{\Omega}$ has a sparse inverse, task pairs corresponding to zero entries in $\boldsymbol{\Omega}^{-1}$ will not be explicitly coupled in the penalty of (6). Similarly, a zero entry in $\boldsymbol{\Sigma}^{-1}$ indicates no direct interaction between the two corresponding features in the penalty. Also, note that a clustering of tasks can be expressed by block-wise sparsity of $\boldsymbol{\Omega}^{-1}$.

Covariance selection aims to select nonzero entries in the Gaussian inverse covariance and discover conditional independence between variables (indicated by zero entries in the inverse covariance) [14, 5, 17, 15]. The matrix-normal density in eq. (6) enables us to perform sparse covariance selection to regularize and select task and feature structures.

Formally, we rewrite (6) to include two additional $\ell 1$ penalty terms on the inverse covariances:

$$\mathcal{L} = \sum_{t=1}^{m} \sum_{i=1}^{n_t} L(y_i^{(t)}, \mathbf{x}_i^{(t)}, \mathbf{W}(t,:)) + \lambda [p \log |\boldsymbol{\Omega}| + m \log |\boldsymbol{\Sigma}| + tr\{\boldsymbol{\Omega}^{-1} \mathbf{W} \boldsymbol{\Sigma}^{-1} \mathbf{W}^T\}]$$
$$+ \lambda_\Omega ||\boldsymbol{\Omega}^{-1}||_{\ell 1} + \lambda_\Sigma ||\boldsymbol{\Sigma}^{-1}||_{\ell 1} \quad (9)$$

where $|| \; ||_{\ell 1}$ is the $\ell 1$-norm of a matrix, and $\lambda_\Omega$ and $\lambda_\Sigma$ control the strength of $\ell 1$ penalties and therefore the sparsity of task and feature structures.

Based on the new regularization formula (9), estimating $\mathbf{W}$ given $\boldsymbol{\Omega}$ and $\boldsymbol{\Sigma}$ as in (5) is not affected, while inferring $\boldsymbol{\Omega}$ and $\boldsymbol{\Sigma}$ given $\mathbf{W}$, previously shown as (7), becomes a new problem:

$$\min_{\boldsymbol{\Omega}, \boldsymbol{\Sigma}} \; p \log |\boldsymbol{\Omega}| + m \log |\boldsymbol{\Sigma}| + tr\{\boldsymbol{\Omega}^{-1} \mathbf{W} \boldsymbol{\Sigma}^{-1} \mathbf{W}^T\} + \frac{\lambda_\Omega}{\lambda} ||\boldsymbol{\Omega}^{-1}||_{\ell 1} + \frac{\lambda_\Sigma}{\lambda} ||\boldsymbol{\Sigma}^{-1}||_{\ell 1} \quad (10)$$

As in (8), we can iteratively optimize $\boldsymbol{\Omega}$ and $\boldsymbol{\Sigma}$ until convergence, as follows:

$$\left\{ \begin{array}{rcl} \hat{\boldsymbol{\Omega}} &=& \text{argmin}_{\boldsymbol{\Omega}} \; p \log |\boldsymbol{\Omega}| + tr\{\boldsymbol{\Omega}^{-1} (\mathbf{W} \boldsymbol{\Sigma}^{-1} \mathbf{W}^T)\} + \frac{\lambda_\Omega}{\lambda} ||\boldsymbol{\Omega}^{-1}||_{\ell 1} \\ \hat{\boldsymbol{\Sigma}} &=& \text{argmin}_{\boldsymbol{\Sigma}} \; m \log |\boldsymbol{\Sigma}| + tr\{\boldsymbol{\Sigma}^{-1} (\mathbf{W}^T \hat{\boldsymbol{\Omega}}^{-1} \mathbf{W})\} + \frac{\lambda_\Sigma}{\lambda} ||\boldsymbol{\Sigma}^{-1}||_{\ell 1} \end{array} \right. \quad (11)$$

Note that both equations in (11) are $\ell 1$ regularized covariance selection problems, for which efficient optimization has been intensively studied [5, 17, 15]. For example, we can use graphical lasso [17] as a basic solver and consider (11) as an $\ell 1$ regularized "flip-flop" algorithm:

$$\left\{ \begin{array}{rcl} \hat{\boldsymbol{\Omega}} &=& glasso(\frac{1}{p} \mathbf{W} \hat{\boldsymbol{\Sigma}}^{-1} \mathbf{W}^T, \frac{\lambda_\Omega}{\lambda}) \\ \hat{\boldsymbol{\Sigma}} &=& glasso(\frac{1}{m} \mathbf{W}^T \hat{\boldsymbol{\Omega}}^{-1} \mathbf{W}, \frac{\lambda_\Sigma}{\lambda}) \end{array} \right.$$

Finally, an annoying part of eq. (9) is the presence of two additional regularization parameters $\lambda_\Omega$ and $\lambda_\Sigma$. Due to the property of matrix normal distributions that only $\Sigma \otimes \Omega$ is identifiable, we can safely reduce the complexity of choosing regularization parameters by considering the restriction:

$$\lambda_\Omega = \lambda_\Sigma \tag{12}$$

The following lemma proves that restricting $\lambda_\Omega$ and $\lambda_\Sigma$ to be equal in eq. (9) will not reduce the space of optimal models $\mathbf{W}$ we can obtain. As a result, we eliminate one regularization parameter.

**Lemma 1.** Suppose $\mathbf{W}^*$ belongs to a minimizer $(\mathbf{W}^*, \mathbf{\Omega}^*, \mathbf{\Sigma}^*)$ for eq. (9) with some arbitrary choice of $\lambda$, $\lambda_\Omega$ and $\lambda_\Sigma > 0$. Then, $\mathbf{W}^*$ must also belong to a minimizer for eq. (9) with certain choice of $\lambda'$, $\lambda'_\Omega$ and $\lambda'_\Sigma$ such that $\lambda'_\Omega = \lambda'_\Sigma$. Proof of lemma 1 is provided in Appendix A.

### 4.3 The Algorithm

Based on the regularization formula (9), we study the following algorithm to learning multiple tasks:

1) Estimate $\mathbf{W}$ by solving (5), using $\mathbf{\Omega} = \mathbf{I}_m$ and $\mathbf{\Sigma} = \mathbf{I}_p$;

2) Infer $\mathbf{\Omega}$ and $\mathbf{\Sigma}$ in (9) (by solving (11) until convergence), using the estimated $\mathbf{W}$ from step 1);

3) Estimate $\mathbf{W}$ by solving (5), using the inferred $\mathbf{\Omega}$ and $\mathbf{\Sigma}$ from step 2).

One can safely iterate over steps 2) and 3) and convergence to a local minimum of eq. (9) is guaranteed. However, we observed that a single pass yields good results[1]. Steps 1) and 3) are linear in the number of data points and step 2) is independent of it, so the method scales well with the number of samples. Step 2) needs to solve $\ell 1$ regularized covariance selection problems as (11). We use the state of the art technique [17], but more efficient optimization for large covariances is still desirable.

### 4.4 Additional Constraints

We can have additional structure assumptions in the matrix-normal penalty. For example, consider:

$$
\begin{align}
\Omega_{ii} &= 1 \quad i = 1, 2, \ldots, m \tag{13} \\
\Sigma_{jj} &= 1 \quad j = 1, 2, \ldots, p \tag{14}
\end{align}
$$

In this case, we ignore variances and restrict our attention to *correlation* structures. For example, off-diagonal entries of task covariance $\mathbf{\Omega}$ characterize the task similarity; diagonal entries indicate different amounts of regularization on tasks, which may be fixed as a constant if we prefer tasks to be equally regularized. Similar arguments apply to feature covariance $\mathbf{\Sigma}$. We include these restrictions by converting inferred covariance(s) into correlation(s) in step 2) of the algorithm in Section 4.3. In other words, the restrictions are enforced by a projection step.

If one wants to iterative over steps 2) and 3) of the algorithm in Section 4.3 until convergence, we may consider the constraints

$$
\begin{align}
\Omega_{ii} &= c_1 \quad i = 1, 2, \ldots, m \tag{15} \\
\Sigma_{jj} &= c_2 \quad j = 1, 2, \ldots, p \tag{16}
\end{align}
$$

with unknown quantities $c_1$ and $c_2$, and consider eq. (9) in step 2) as a *constrained optimization* problem w.r.t. $\mathbf{W}$, $\mathbf{\Omega}$, $\mathbf{\Sigma}$, $c_1$ and $c_2$, instead of using a projection step. As a result, the "flip-flop" algorithm in (11) needs to solve $\ell 1$ penalized covariance selection with equality constraints (15) or (16), where the dual block coordinate descent [5] and graphical lasso [17] are no longer directly applicable. In this case, one can solve the two steps of (11) as determinant maximization problems with linear constraints [25], but this is inefficient. We will study this direction (efficient constrained sparse covariance selection) in the future work.

## 5 Empirical Studies

In this section, we present our empirical studies on a landmine detection problem and a face recognition problem, where multiple tasks correspond to detecting landmines at different landmine fields and classifying faces between different subjects, respectively.

## 5.1 Data Sets and Experimental Settings

**The landmine detection data set** from [26] contains examples collected from different landmine fields. Each example in the data set is represented by a 9-dimensional feature vector extracted from radar imaging, which includes moment-based features, correlation-based features, an energy ratio feature and a spatial variance feature. As a binary classification problem, the goal is to predict landmines (positive class) or clutter (negative class). Following [26], we jointly learn 19 tasks from landmine fields $1-10$ and $19-24$ in the data set. As a result, the model parameters $\mathbf{W}$ are a $19 \times 10$ matrix, corresponding to 19 tasks and 10 coefficients (including the intercept) for each task.

The distribution of examples is imbalanced in each task, with a few dozen positive examples and several hundred negative examples. Therefore, we use the average AUC (Area Under the ROC Curve) over 19 tasks as the performance measure. We vary the size of the training set for each task as $30, 40, 80$ and $160$. Note that we intentionally keep the training sets small because the need for cross-task learning diminishes as the training set becomes large relative to the number of parameters being learned. For each training set size, we randomly select training examples for each task and the rest is used as the testing set. This is repeated 30 times. Task-average AUC scores are collected over 30 runs, and mean and standard errors are reported. Note that for small training sizes (e.g., 30 per task) we often have some task(s) that do *not* have any positive training sample. It is interesting to see how well multi-task learning handles this case.

**The face recognition data set** is the Yale face database, which contains 165 images of 15 subjects. The 11 images per subject correspond to different configurations in terms of expression, emotion, illumination, and wearing glasses (or not), etc. Each image is scaled to $32 \times 32$ pixels. We use the first 8 subjects to construct $\frac{8 \times 7}{2} = 28$ binary classification tasks, each to classify two subjects. We vary the size of the training set as $3, 5$ and $7$ images per subject. We have 30 random runs for each training size. In each run, we randomly select the training set and use the rest as the testing set. We collect task-average classification errors over 30 runs, and report mean and standard errors.

Choice of features is important for face recognition problems. In our experiments, we use orthogonal Laplacianfaces [10], which have been shown to provide better discriminative power than Eigenfaces (PCA), fisherfaces (LDA) and Laplacianfaces on several benchmark data sets. In each random run, we extract 30 orthogonal Laplacianfaces using the selected training set of all 8 subjects[2], and conduct experiments of all 28 classification tasks in the extracted feature space.

## 5.2 Models and Implementation Details

We use the logistic regression loss as the empirical loss $L$ in (9). We compare the following models.

**STL**: learn $\ell 2$ regularized logistic regression for each task separately.

**MTL-C**: clustered multi-task learning [21], which encourages task clustering in regularization. As discussed in Section 4.1, this is related to eq. (5) with only a task structure $\mathbf{\Omega}$.

**MTL-F**: multi-task feature learning [2], which corresponds to fixing the task covariance $\mathbf{\Omega}$ as $\mathbf{I}_m$ and optimizing (6) with only the feature covariance $\mathbf{\Sigma}$.

In addition, we also study various different configurations of the proposed framework:

**MTL($\mathbf{I}_m$&$\mathbf{I}_p$)**: learn $\mathbf{W}$ using (9) with $\mathbf{\Omega}$ and $\mathbf{\Sigma}$ fixed as identity matrices $\mathbf{I}_m$ and $\mathbf{I}_p$.

**MTL($\mathbf{\Omega}$&$\mathbf{I}_p$)**: learn $\mathbf{W}$ and task covariance $\mathbf{\Omega}$ using (9), with feature covariance $\mathbf{\Sigma}$ fixed as $\mathbf{I}_p$.

**MTL($\mathbf{I}_m$&$\mathbf{\Sigma}$)**: learn $\mathbf{W}$ and feature covariance $\mathbf{\Sigma}$ using (9), with task covariance $\mathbf{\Omega}$ fixed as $\mathbf{I}_m$.

**MTL($\mathbf{\Omega}$&$\mathbf{\Sigma}$)**: learn $\mathbf{W}, \mathbf{\Omega}$ and $\mathbf{\Sigma}$ using (9), inferring both task and feature structures.

**MTL($\mathbf{\Omega}$&$\mathbf{\Sigma}$)$_{\Omega_{ii}=\Sigma_{jj}=1}$**: learn $\mathbf{W}, \mathbf{\Omega}$ and $\mathbf{\Sigma}$ using (9), with restricted $\mathbf{\Omega}$ and $\mathbf{\Sigma}$ as (13) and (14).

**MTL($\mathbf{\Omega}$&$\mathbf{\Sigma}$)$_{\Omega_{ii}=1}$**: learn $\mathbf{W}, \mathbf{\Omega}$ and $\mathbf{\Sigma}$ using (9), with restricted $\mathbf{\Omega}$ as (13) and free $\mathbf{\Sigma}$. Intuitively, free diagonal entries in $\mathbf{\Sigma}$ are useful when features are of different importance, e.g, components extracted as orthogonal Laplacianfaces usually capture decreasing amounts of information [10].

We use conjugate gradients [22] to optimize $\mathbf{W}$ in (5), and infer $\mathbf{\Omega}$ and $\mathbf{\Sigma}$ in (11) using graphical lasso [17] as the basic solver. Regularization parameters $\lambda$ and $\lambda_{\Omega} = \lambda_{\Sigma}$ are chosen by 3-fold cross

| Avg AUC Score | 30 samples | 40 samples | 80 samples | 160 samples |
|---|---|---|---|---|
| STL | 64.85(0.52) | 67.62(0.64) | 71.86(0.38) | 76.22(0.25) |
| MTL-C [21] | 67.09(0.44) | 68.95(0.40) | 72.89(0.31) | 76.64(0.17) |
| MTL-F [2] | 72.39(0.79) | 74.75(0.63) | 77.12(0.18) | **78.13(0.12)** |
| MTL($\mathbf{I}_m$&$\mathbf{I}_p$) | 66.10(0.65) | 69.91(0.40) | 73.34(0.28) | 76.17(0.22) |
| MTL($\mathbf{\Omega}$&$\mathbf{I}_p$) | **74.88(0.29)** | 75.83(0.28) | 76.93(0.15) | 77.95(0.17) |
| MTL($\mathbf{I}_m$&$\mathbf{\Sigma}$) | 72.71(0.65) | 74.98(0.32) | **77.35(0.14)** | **78.13(0.14)** |
| MTL($\mathbf{\Omega}$&$\mathbf{\Sigma}$) | **75.10(0.27)** | 76.16(0.15) | **77.32(0.24)** | **78.21(0.17)**$^*$ |
| MTL($\mathbf{\Omega}$&$\mathbf{\Sigma}$)$_{\Omega_{ii}=\Sigma_{jj}=1}$ | **75.31(0.26)**$^*$ | **76.64(0.13)**$^*$ | **77.56(0.16)**$^*$ | **78.01(0.12)** |
| MTL($\mathbf{\Omega}$&$\mathbf{\Sigma}$)$_{\Omega_{ii}=1}$ | **75.19(0.22)** | 76.25(0.14) | **77.22(0.15)** | **78.03(0.15)** |

Table 1: Average AUC scores (%) on landmine detection: means (and standard errors) over 30 random runs. For each column, the best model is marked with $^*$ and competitive models (by paired t-tests) are shown in **bold**.

validation within the range $[10^{-7}, 10^3]$. The model in [21] uses 4 regularization parameters, and we consider 3 values for each parameter, leading to $3^4 = 64$ combinations chosen by cross validation.

## 5.3 Results on Landmine Detection

The results on landmine detection are shown in Table 1. Each row of the table corresponds to a model in our experiments. Each column is a training sample size. We have 30 random runs for each sample size. We use task-average AUC score as the performance measure and report the mean and standard error of this measure over 30 random runs. The best model is marked with $^*$, and models displayed in bold fonts are statistically competitive models (i.e. not significantly inferior to the best model in a one-sided paired t-test with $\alpha = 0.05$).

Overall speaking, MTL($\mathbf{\Omega}$&$\mathbf{\Sigma}$) and MTL($\mathbf{\Omega}$&$\mathbf{\Sigma}$)$_{\Omega_{ii}=\Sigma_{jj}=1}$ lead to the best prediction performance. For small training sizes, restricted $\mathbf{\Omega}$ and $\mathbf{\Sigma}$ ($\Omega_{ii} = \Sigma_{jj} = 1$) offer better prediction; for large training size (160 per task), free $\mathbf{\Omega}$ and $\mathbf{\Sigma}$ give the best performance. The best model performs better than MTL-F [2] and much better than MTL-C [21] with small training sets.

MTL($\mathbf{I}_m$&$\mathbf{I}_p$) performs better than STL, i.e., even the simplest coupling among tasks (by sharing $\lambda$) can be helpful when the size of training data is small. Consider the performance of MTL($\mathbf{\Omega}$&$\mathbf{I}_p$) and MTL($\mathbf{I}_m$&$\mathbf{\Sigma}$), which learn either a task structure or a feature structure. When the size of training samples is small (i.e., 30 or 40), coupling by task similarity is more effective, and as the training size increases, learning a common feature representation is more helpful. Finally, consider MTL($\mathbf{\Omega}$&$\mathbf{\Sigma}$), MTL($\mathbf{\Omega}$&$\mathbf{\Sigma}$)$_{\Omega_{ii}=\Sigma_{jj}=1}$ and MTL($\mathbf{\Omega}$&$\mathbf{\Sigma}$)$_{\Omega_{ii}=1}$. MTL($\mathbf{\Omega}$&$\mathbf{\Sigma}$)$_{\Omega_{ii}=\Sigma_{jj}=1}$ imposes a strong restriction and leads to better performance when the training size is small. MTL($\mathbf{\Omega}$&$\mathbf{\Sigma}$) is more flexible and performs well given large numbers of training samples. MTL($\mathbf{\Omega}$&$\mathbf{\Sigma}$)$_{\Omega_{ii}=1}$ performs similarly to MTL($\mathbf{\Omega}$&$\mathbf{\Sigma}$)$_{\Omega_{ii}=\Sigma_{jj}=1}$, indicating no significant variation of feature importance in this problem.

## 5.4 Results on Face Recognition

Empirical results on face recognition are shown in Table 2, with the best model in each column marked with $^*$ and competitive models displayed in bold. MTL-C [21] performs even worse than STL. One possible explanation is that, since tasks are to classify faces between different subjects, there may not be a clustered structure over tasks and thus a cluster norm will be inappropriate. In this case, using a task similarity matrix may be more appropriate than clustering over tasks. In addition, MTL($\mathbf{\Omega}$&$\mathbf{\Sigma}$)$_{\Omega_{ii}=1}$ shows advantages over other models, especially if given relatively sufficient training data (5 or 7 per subject). Compared to MTL($\mathbf{\Omega}$&$\mathbf{\Sigma}$), MTL($\mathbf{\Omega}$&$\mathbf{\Sigma}$)$_{\Omega_{ii}=1}$ imposes restrictions on diagonal entries of task covariance $\mathbf{\Omega}$: all tasks seem to be similarly difficult and should be equally regularized. Compared to MTL($\mathbf{\Omega}$&$\mathbf{\Sigma}$)$_{\Omega_{ii}=\Sigma_{jj}=1}$, MTL($\mathbf{\Omega}$&$\mathbf{\Sigma}$)$_{\Omega_{ii}=1}$ allows the diagonal entries of feature covariance $\mathbf{\Sigma}$ to capture varying degrees of importance of Laplacianfaces.

| Avg Classification Errors | 3 samples per class | 5 samples per class | 7 samples per class |
|---|---|---|---|
| STL | 10.97(0.46) | 7.62(0.30) | 4.75(0.35) |
| MTL-C [21] | 11.09(0.49) | 7.87(0.34) | 5.33(0.34) |
| MTL-F [2] | 10.78(0.60) | 6.86(0.27) | 4.20(0.31) |
| MTL($\mathbf{I}_m$&$\mathbf{I}_p$) | 10.88(0.48) | 7.51(0.28) | 5.00(0.35) |
| MTL($\mathbf{\Omega}$&$\mathbf{I}_p$) | 9.98(0.55) | 6.68(0.30) | 4.12(0.38) |
| MTL($\mathbf{I}_m$&$\mathbf{\Sigma}$) | **9.87(0.59)** | **6.25(0.27)** | 4.06(0.34) |
| MTL($\mathbf{\Omega}$&$\mathbf{\Sigma}$) | **9.81(0.49)** | **6.23(0.29)** | 4.11(0.36) |
| MTL($\mathbf{\Omega}$&$\mathbf{\Sigma}$)$_{\Omega_{ii}=\Sigma_{jj}=1}$ | **9.67(0.57)**$^*$ | **6.21(0.28)** | 4.02(0.32) |
| MTL($\mathbf{\Omega}$&$\mathbf{\Sigma}$)$_{\Omega_{ii}=1}$ | **9.67(0.51)**$^*$ | **5.98(0.29)**$^*$ | **3.53(0.34)**$^*$ |

Table 2: Average classification errors (%) on face recognition: means (and standard errors) over 30 random runs. For each column, the best model is marked with $^*$ and competitive models (by paired t-tests) are shown in **bold**.

# 6 Conclusion

We propose a matrix-variate normal penalty with sparse inverse covariances to couple multiple tasks. The proposed framework provides an effective and flexible way to characterize and select both task and feature structures for learning multiple tasks. Several recently proposed methods can be viewed as variants of the special cases of our formulation and our empirical results on landmine detection and face recognition show that we consistently outperform previous methods.

**Acknowledgement**: this work was funded in part by the National Science Foundation under grant NSF-IIS0911032 and the Department of Energy under grant DESC0002607.

# Appendix A

**Proof of Lemma 1.**

We prove lemma 1 by construction. Given an arbitrary choice of $\lambda$, $\lambda_\Omega$ and $\lambda_\Sigma > 0$ in eq. (9) and an optimal solution $(\mathbf{W}^*, \mathbf{\Omega}^*, \mathbf{\Sigma}^*)$, we want to prove that $\mathbf{W}^*$ also belongs to an optimal solution for eq. (9) with certain $\lambda'$, $\lambda'_\Omega$ and $\lambda'_\Sigma$ s.t. $\lambda'_\Omega = \lambda'_\Sigma$. Let's construct $\lambda'$, $\lambda'_\Omega$ and $\lambda'_\Sigma$ as follows:

$$(\lambda', \lambda'_\Omega, \lambda'_\Sigma) = (\lambda, \sqrt{\lambda_\Omega \lambda_\Sigma}, \sqrt{\lambda_\Omega \lambda_\Sigma}) \tag{17}$$

We denote the objective function in eq. (9) with $\lambda$, $\lambda_\Omega$ and $\lambda_\Sigma$ as $Obj^{\lambda, \lambda_\Omega, \lambda_\Sigma}(\mathbf{W}, \mathbf{\Omega}, \mathbf{\Sigma})$. Also, we denote the objective function with our constructed parameters $\lambda'$, $\lambda'_\Omega$ and $\lambda'_\Sigma$ as $Obj^{\lambda', \lambda'_\Omega, \lambda'_\Sigma}(\mathbf{W}, \mathbf{\Omega}, \mathbf{\Sigma})$.

For any $(\mathbf{W}, \mathbf{\Omega}, \mathbf{\Sigma})$, we further construct an invertible (i.e., one-to-one) transform as follows:

$$(\mathbf{W}', \mathbf{\Omega}', \mathbf{\Sigma}') = (\mathbf{W}, \sqrt{\frac{\lambda_\Sigma}{\lambda_\Omega}}\mathbf{\Omega}, \sqrt{\frac{\lambda_\Omega}{\lambda_\Sigma}}\mathbf{\Sigma}) \tag{18}$$

The key step in our proof is that, by construction, the following equality always holds:

$$Obj^{\lambda, \lambda_\Omega, \lambda_\Sigma}(\mathbf{W}, \mathbf{\Omega}, \mathbf{\Sigma}) = Obj^{\lambda', \lambda'_\Omega, \lambda'_\Sigma}(\mathbf{W}', \mathbf{\Omega}', \mathbf{\Sigma}') \tag{19}$$

To see this, notice that eq. (9) consists of three parts. The first part is the empirical loss on training examples, depending only on $\mathbf{W}$ (and training data). The second part is the log-density of matrix normal distributions, which depends on $\mathbf{W}$ and $\mathbf{\Sigma} \otimes \mathbf{\Omega}$. The third part is the sum of two $\ell 1$ penalties. The equality in eq. (19) stems from the fact that all three parts of eq. (9) are not changed: 1) $\mathbf{W}' = \mathbf{W}$ so the first part remains unchanged; 2) $\mathbf{\Sigma}' \otimes \mathbf{\Omega}' = \mathbf{\Sigma} \otimes \mathbf{\Omega}$ so the second part of the matrix normal log-density is the same; 3) by our construction, the third part is not changed.

Based on this equality, if $(\mathbf{W}^*, \mathbf{\Omega}^*, \mathbf{\Sigma}^*)$ minimizes $Obj^{\lambda, \lambda_\Omega, \lambda_\Sigma}()$, we have that $(\mathbf{W}^*, \sqrt{\frac{\lambda_\Sigma}{\lambda_\Omega}}\mathbf{\Omega}^*, \sqrt{\frac{\lambda_\Omega}{\lambda_\Sigma}}\mathbf{\Sigma}^*)$ minimizes $Obj^{\lambda', \lambda'_\Omega, \lambda'_\Sigma}()$, where $\lambda' = \lambda$ and $\lambda'_\Omega = \lambda'_\Sigma = \sqrt{\lambda_\Omega \lambda_\Sigma}$.

## Footnotes

[1]Further iterations over step 2) and 3) will not dramatically change model estimation. Also, early stopping as regularization might also lead to better generalizability.

[2]For experiments with 3 images per subject, we can only extract 23 Laplacianfaces, which is limited by the size of training examples ($3 \times 8 = 24$) [10].

# References

[1] R. K. Ando and T. Zhang. A framework for learning predictive structures from multiple tasks and unlabeled data. *Journal of Machine Learning Research*, 6:1817–1853, 2005.

[2] A. Argyriou, T. Evgeniou, and M. Pontil. Multi-task feature learning. In *NIPS*, 2006.

[3] A. Argyriou, C. A. Micchelli, M. Pontil, and Y. Ying. A spectral regularization framework for multi-task structure learning. In *NIPS*, 2007.

[4] B. Bakker and T. Heskes. Task clustering and gating for bayesian multitask learning. *Journal of Machine Learning Research*, 4:83–99, 2003.

[5] O. Banerjee, L. E. Ghaoui, and A. d'Aspremont. Model selection through sparse maximum likelihood estimation for multivariate gaussian or binary data. *J. Mach. Learn. Res.*, 9:485–516, 2008.

[6] J. Baxter. Learning Internal Representations. In *COLT*, pages 311–320, 1995.

[7] E. Bonilla, K. M. Chai, and C. Williams. Multi-task gaussian process prediction. In J. Platt, D. Koller, Y. Singer, and S. Roweis, editors, *NIPS*, pages 153–160. 2008.

[8] E. V. Bonilla, F. V. Agakov, and C. K. I. Williams. Kernel multi-task learning using task-specific features. In *AISTATS*, 2007.

[9] P. J. Brown and M. Vannucci. Multivariate Bayesian Variable Selection and Prediction. *Journal of the Royal Statistical Soceity, Series B*, 60(3):627–641, 1998.

[10] D. Cai, X. He, J. Han, and H. Zhang. Orthogonal laplacianfaces for face recognition. *IEEE Transactions on Image Processing*, 15(11):3608–3614, 2006.

[11] R. Caruana. Multitask Learning. *Machine Learning*, 28:41–75, 1997.

[12] J. Chen, L. Tang, J. Liu, and J. Ye. A Convex Formulation for Learning Shared Structures from Multiple Tasks. In *ICML*, 2009.

[13] A. P. Dawid. Some matrix-variate distribution theory: Notational considerations and a bayesian application. *Biometrika*, 68(1):265–274, 1981.

[14] A. P. Dempster. Covariance selection. *Biometrics*, 1972.

[15] J. Duchi, S. Gould, and D. Koller. Projected subgradient methods for learning sparse gaussians. In *Proceedings of the Twenty-fourth Conference on Uncertainty in AI (UAI)*, 2008.

[16] P. Dutilleul. The MLE Algorithm for the Matrix Normal Distribution. *J. Statist. Comput. Simul.*, 64:105–123, 1999.

[17] J. Friedman, T. Hastie, and R. Tibshirani. Sparse inverse covariance estimation with the graphical lasso. *Biostatistics*, 2007.

[18] A. K. Gupta and D. K. Nagar. *Matrix Variate Distributions*. Chapman Hall, 1999.

[19] B. Hariharan, S. Vishwanathan, and M. Varma. Large scale max-margin multi-label classification with priors. In *ICML*, 2010.

[20] T. Hastie, R. Tibshirani, and J. Friedman. *The Elements of Statistical Learning: Data Mining, Inference, and Prediction*. Springer, 2001.

[21] L. Jacob, F. Bach, and J. P. Vert. Clustered multi-task learning: A convex formulation. In *NIPS*, pages 745–752, 2008.

[22] J. Nocedal and S. Wright. *Numerical Optimization*. Springer, 2000.

[23] G. Obozinski, B. Taskar, and M. I. Jordan. Joint covariate selection and joint subspace selection for multiple classification problems. *Statistics and Computing*, 2009.

[24] S. Thrun and J. O'Sullivan. Discovering Structure in Multiple Learning Tasks: The TC Algorithm. In *ICML*, pages 489–497, 1996.

[25] L. Vandenberghe, S. Boyd, and S.-P. Wu. Determinant maximization with linear matrix inequality constraints. *SIAM Journal on Matrix Analysis and Applications*, 19:499–533, 1996.

[26] Y. Xue, X. Liao, L. Carin, and B. Krishnapuram. Multi-task learning for classification with dirichlet process priors. *Journal of Machine Learning Research*, 8:35–63, 2007.

[27] K. Yu, W. Chu, S. Yu, V. Tresp, and Z. Xu. Stochastic relational models for discriminative link prediction. In *NIPS*, pages 1553–1560, 2007.

[28] K. Yu, J. Lafferty, S. Zhu, and Y. Gong. Large-scale collaborative prediction using a nonparametric random effects model. In *ICML*, pages 1185–1192, 2009.

[29] S. Yu, V. Tresp, and K. Yu. Robust multi-task learning with t-processes. In *ICML*, page 1103, 2007.

[30] J. Zhang, Z. Ghahramani, and Y. Yang. Learning multiple related tasks using latent independent component analysis. In *NIPS*, pages 1585–1592, 2006.

